# Field-Programmable Learning Arrays

**Seth Bridges, Miguel Figueroa, David Hsu, and Chris Diorio**
Department of Computer Science and Engineering
University of Washington
114 Sieg Hall, Box 352350
Seattle, WA 98195-2350
{*seth,miguel,hsud,diorio*}*@cs.washington.edu*

## Abstract

This paper introduces the Field-Programmable Learning Array, a new paradigm for rapid prototyping of learning primitives and machine-learning algorithms in silicon. The FPLA is a mixed-signal counterpart to the all-digital Field-Programmable Gate Array in that it enables rapid prototyping of algorithms in hardware. Unlike the FPGA, the FPLA is targeted directly for machine learning by providing local, parallel, on-line analog learning using floating-gate MOS synapse transistors. We present a prototype FPLA chip comprising an array of reconfigurable computational blocks and local interconnect. We demonstrate the viability of this architecture by mapping several learning circuits onto the prototype chip.

## 1 Introduction

Implementing machine-learning algorithms in VLSI is a logical step toward enabling real-time or mobile applications of these algorithms [1]. Several machine-learning architectures such as *neural networks* and *Bayes nets* map naturally to VLSI, because each uses many simple elements in parallel and computes using only local information. Such algorithms, when implemented in VLSI, can leverage the inherent parallelism offered by the millions of transistors on a single silicon die. Depending on the design technique, hardware implementations of learning algorithms can realize significant performance increases over standard computers in terms of speed or power consumption.

Despite the benefits of implementing machine-learning algorithms in VLSI, several issues have kept hardware implementations from penetrating mainstream machine learning. First, many previous hardware systems were not scalable due to the size of many primary components such as digital multipliers or digital-to-analog converters[2, 3]. Second, many systems such as [4] have inflexible circuit topologies, allowing them to be used for only very specific problems. Third, many hardware learning systems did not comprise a complete solution with on-chip learning [5] and often required external weight updates[3, 6]. In addition to these problems of scalability and inflexibility, perhaps the biggest impediment to implementing learning in VLSI is that designing VLSI chips is a time-consuming and error-prone process. All current VLSI learning implementations required a detailed knowledge of analog and digital circuit design. This prerequisite knowledge impedes hardware development by a hardware novice; indeed, the design process can challenge even the most

experienced circuit designer. Because we make extensive use of floating-gate synapse transistors [1] in our learning circuits to enable local adaptation, the design process becomes even more difficult due to slow and inaccurate simulation of these devices.

A reconfigurable learning system would solve these problems by allowing rapid prototyping and flexibility in learning system hardware. Also, reconfigurability allows the system to adapt to changes in the problem definition. For example, a designer can trade input dimensionality for resolution by reallocating FPLA resources, even after the implementation is complete. A custom VLSI solution would not allow such tradeoffs after fabrication. When combined with a simple user interface, a reconfigurable learning system can enable anyone with a machine-learning background to express his/her ideas in hardware.

In this paper, we propose a mixed analog-digital Field-Programmable Learning Array (FPLA), a reconfigurable system for rapid prototyping of machine-learning algorithms in hardware. The FPLA enables the design cycle shown in Figure 1(a) in which the designer expresses a machine-learning problem as an algorithm, compiles that representation into an FPLA configuration, and prototypes the algorithm in an FPLA. The FPLA is similar in concept to all-digital Field-Programmable Gate Arrays (FPGA), in that they both enable reconfigurable computation and prototyping using arrays of simple elements and reconfigurable wiring. Unlike previous reconfigurable hardware learning solutions [3, 4, 6, 7], the FPLA is a general-purpose prototyping tool and does not target one specific architecture. Moreover, our FPLA supports on-chip adaptation and enables rapid prototyping of a large class of learning algorithms.

We have implemented a prototype core for an FPLA. Our chip comprises a small ($2 \times 2$) array of Programmable Learning Blocks (PLBs) as well as a simple interconnect structure to allow the PLBs to communicate in an all-to-all fashion. Our results show that this prototype system achieves its design goal of enabling rapid prototyping of floating-gate learning circuits by implementing learning circuits known in the literature as well as new circuits prototyped for the first time.

The remainder of the paper proceeds as follows. In section 2, we discuss the proposed FPLA architecture, as well as the subset that is our prototype. Section 3 shows results from our test chip of the prototype design. Section 4 concludes with a discussion of improvements that we are making to the design and opportunities for future work.

## 2   FPLA Architecture

### 2.1   An FPLA Architecture

Our proposed FPLA architecture, shown in Figure 1(b), has three properties that enable machine learning: 1) a core comprising an array of Programmable Learning Blocks to compute machine-learning functions, 2) reconfigurable interconnect to enable inter-PLB communication, 3) the ability to compute with sufficient accuracy, and 4) a simple and well-defined user interface.

The first two properties are dimensions of the FPLA design space, where tradeoffs between them results in varying levels of flexibility and functionality at the cost of area and power. The FPLA core determines the system's functionality. For example, in a task-oriented FPLA, the PLBs that compose the core should allow high-level functions such as multiplication and outer-product learning. Likewise, to develop new learning algorithms in silicon, the PLBs should allow lower-level functions such as current mirrors, differential pairs, and current sources.

In addition to a multi-functional core, a reconfigurable learning array requires flexible interconnect that provides good local connectivity between neighboring PLBs and global

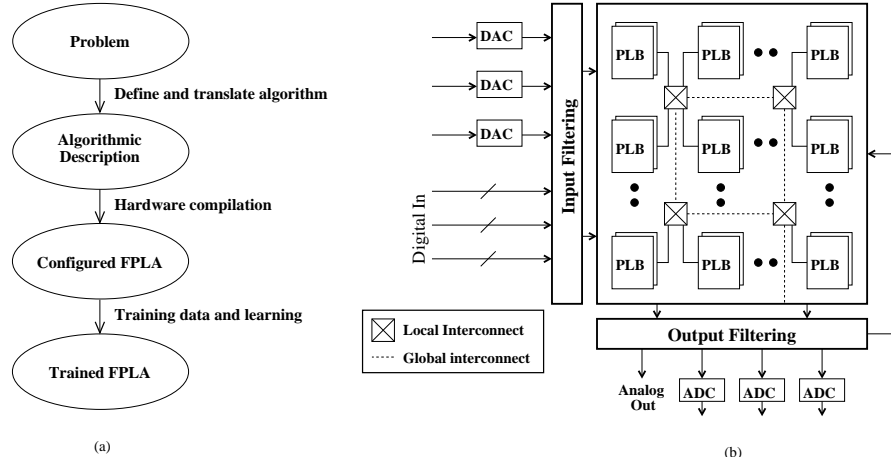

(a)

(b)

Figure 1: (a) FPLA-Based Design Flow. A user programs a machine-learning algorithm and tests it using standard software tools (e.g. Matlab). The design compiler transforms this code into an FPLA configuration, which is then downloaded to the chip. At this point, the FPLA runs the algorithm on a training data set and performs on-chip learning. (b) Proposed FPLA Architecture. The architecture comprises an array of Programmable Learning Blocks (PLBs), a flexible interconnect, and support circuitry on the periphery. Local interconnect enables efficient, low-cost communication between adjacent PLBs. Global interconnect enables distant PLBs to communicate, albeit at a higher cost.

interconnect for long-range connections. The global interconnect must be sparse because of area constraints in VLSI chips, but flexible enough to allow a wide range of PLB connectivity. Local connectivity is critical to enable the creation of complex learning primitives from combinations of PLBs and the implementation of large classes of machine-learning algorithms that exhibit strong local computation.

Analog and mixed signal VLSI systems are typically plagued by offsets and device mismatch. Even though accurate systems are possible[8], the accuracy usually comes at the cost of increased power consumption and die area. The adaptive properties of floating-gate transistors can overcome these intrinsic accuracy limitations[9], therefore enabling mixed analog-digital computation to obtain the best combination of power, area, scalability, and performance.

A user interface for an FPLA comprises two different components: a design compilation and configuration tool, and a chip interface that provides both digital and analog I/O. An FPLA design compiler allows a user to compile an abstract expression of an algorithm (e.g. Matlab code) to an FPLA configuration. The chip interface provides digital I/O to interface with standard computers and surrounding digital circuitry, as well as analog I/O to interface with signals from sensors such as vision chips and implantable devices.

## 2.2 Prototype Chip

As a first step in designing an FPLA, we built a prototype focusing on the PLB design and local interconnect. Our design comprises a 2×2 array of PLBs interconnected in an all-to-all fashion. The system I/O comprises digital input for programming and bidirectional analog input/output for system operation. We show the prototype FPLA architecture and chip micrograph in Figure 2. We fabricated the chip in the TSMC $0.35\mu$m double-poly, four metal process available from MOSIS. The FPLA included two pFET PLBs and two nFET PLBs, each containing 8 uncommitted lines, 4 I/O blocks, and the computational primitives described below. The FPLA occupies $2000\mu$m$\times700\mu$m including the programming 4-

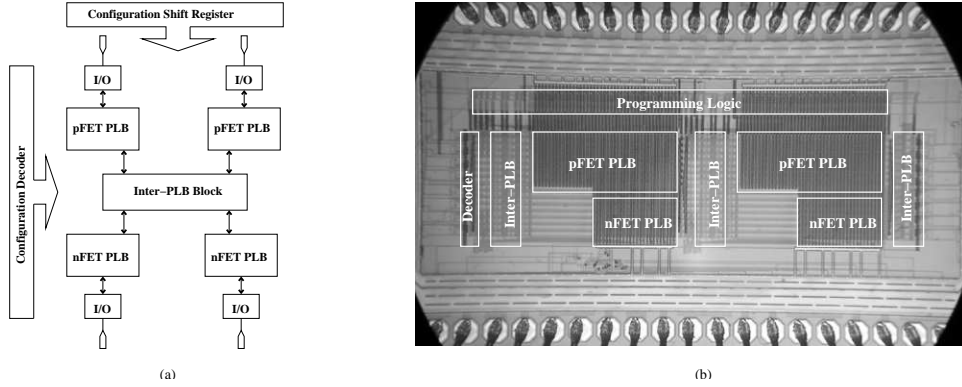

Figure 2: (a) Fabricated Chip Architecture. Our prototype FPLA comprises 4 PLBs that contain simple analog functional primitives. A set of interconnect switches connect the PLBs in an all-to-all fashion. (b) Chip Micrograph. The chip photograph shows the four PLBs, inter-PLB blocks, and programming circuitry. The chip was fabricated in the TSMC $0.35\mu$m double-poly four-metal process from MOSIS.

to-16 decoder and 108-bit shift register. Through design optimization, we have recently reduced the size by more than 50%.

Each of the four PLBs comprises computational circuitry and a large switching matrix built of pass-gates controlled by SRAM. There are two different types of PLBs, the pFET PLB and the nFET PLB, because nFET and pFET are the two flavors of transistors available in standard CMOS processes. The computational primitives that compose the PLBs are two floating-gate transistors, a differential pair, a current mirror, a diode-connected transistor, a bias current source, three transistors with configurable length and width, and two configurable capacitors. These circuit primitives can be wired into arbitrary configurations simply by changing the state of the PLB switch matrix. When deciding what functions to place in the PLBs, our starting point was the decomposition of known primitives [10, 11] for silicon learning as well as standard analog primitives such as those in Mead's book on silicon neural systems [12]. The circuits included in our PLBs are the most common subcircuits found when decomposing these primitives.

Each of the four PLBs is independent of the others and can be programmed and operated independently. However, more useful circuits require resources from multiple PLBs. Inter-PLB blocks provide local connectivity between PLBs where each inter-PLB block is an array of SRAM pass-gate switches that can connect an uncommitted line in one PLB to an uncommitted line in another PLB. The six inter-PLB blocks provide a path from one PLB to any other PLB in the system. To interface with the external world, there are four I/O connections per PLB, each of which can be configured in one of two ways: as a bare connection to the pad for voltage inputs or current outputs, or as a voltage output through a unity-gain buffer. The user configures the FPLA by shifting the configuration bits into the configuration SRAM, located throughout the PLBs and interconnect.

## 3   Implementing Machine-Learning Primitives

To show the correct functionality of our chip, we implemented various circuits from the literature as well as new circuits developed entirely in the FPLA. In the following section, we show results for three of these circuits.

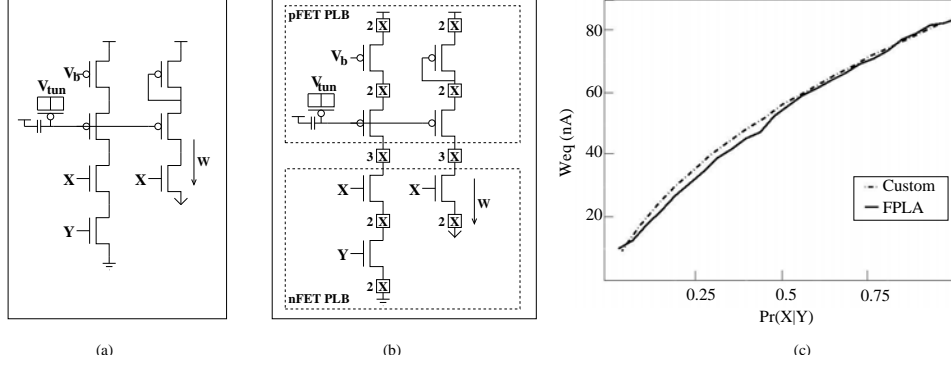

Figure 3: (a) Schematic of the correlational-learning circuit described by Shon and Hsu in [11]. (b) Schematic of the same circuit as implemented in the FPLA. (c) Experimental results comparing the performance of the custom circuit against the reconfigurable circuit. We scaled the data to compensate for differences in operating point between the two implementations. The data reported by Shon and Hsu is smoother because it is averaged over a larger number of experiments.

### 3.1 Correlational-Learning Primitive

As a first test of our chip, we implemented the correlational-learning circuit described by Shon and Hsu in [11]. This circuit learns the conditional probability of a binary event $X$ given another binary event $Y$. We show the original circuit in Figure 3(a), and the FPLA implementation Figure 3(b).

We implemented this circuit using primitives from two PLBs. We input the signals $X$ and $Y$ as voltage pulses. Figure 3(c) compares the results from the custom chip to the results from the FPLA. Both sets of data can be fit by:

$$W_{eq} = I_0 \left( \frac{I_{inj0}}{I_{tun0}} Pr(X|Y) \right)^{\alpha} \tag{1}$$

where $I_0$, $I_{tun0}$, $I_{inj0}$, and $\alpha$ are fit constants. We conclude from this experiment that the correlational-learning circuit, when implemented in the FPLA, operates as the original circuit. SPICE simulations confirm that the interconnect switches have a negligible effect on circuit performance.

### 3.2 Regression-Learning Primitive

The regression-learning circuit described in this section is a new hardware learning primitive first implemented in the FPLA. The circuit performs regression learning on a set of 2-D input data. It comprises two correlational learning circuits like the one shown in Figure 4(a) to encode a differential weight $w$. Each circuit learns $w_+$ and $w_-$ respectively, such that:

$$w = w_+ - w_- \tag{2}$$

The circuit operates as follows. We apply a zero-mean input signal $i$, encoded as a varying current $x$ plus some DC bias current $b$, to the two inputs of the circuit. The differential output current $out$ of each circuit represents the product of its stored weight with the input current.

$$out_+ = (x + b)w_+ \tag{3}$$
$$out_- = (x + b)w_- \tag{4}$$

The difference in those output currents represents the total product of the current input and the weight stored on the floating gate.

$$out = out_+ - out_- = x(w_+ - w_-) + b(w_+ - w_-) \tag{5}$$

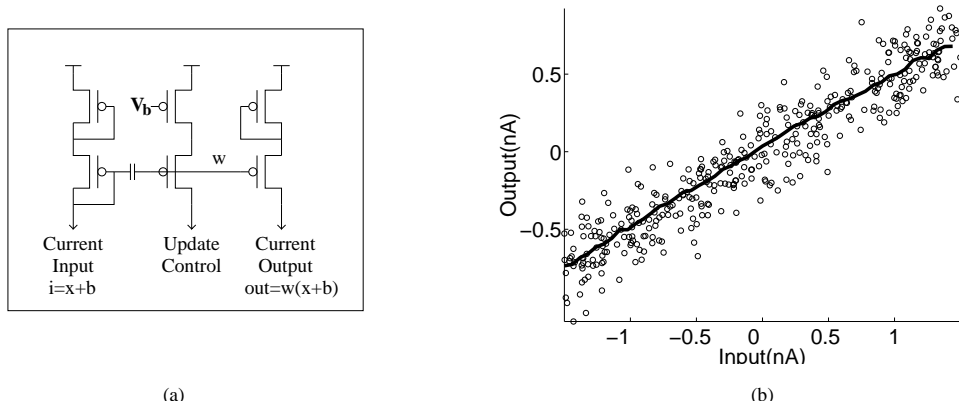

(a)                                                           (b)

Figure 4: (a) Regression Learning Circuit. This circuit is one-half of the regression learning circuit and learns the positive weight $w_+$. The other half of the circuit is identical but used to represent the negative differential weight $w_-$. The difference between the learned weights $w_+$ and $w_-$ converges to the slope of the incoming data. (b) Experimental Data. This data is taken from the FPLA configured as the circuit on the left. The circuit was shown 388 data points with a slope of 0.5 and zero-mean Gaussian noise of 5%. The circuit learned a slope of 0.4924.

where the multiplication is performed by the current mirror formed by the input diode and the floating gate. The output prediction we seek is $wx$, so we remove the scaled input offset current $wb$ with a high-pass filter implemented in the test computer.

$$out_{highpass} = x(w_+ - w_-) \tag{6}$$

Circuit training occurs in a supervised manner. An input $x$ is provided to the circuit, and the circuit predicts an output $wx$. The computer running the test compares that predicted output with the target and feeds an error signal back to the chip. Based on the error signal, the circuit adapts the weight $w$. Positive changes in $w_+$ increase $w$, while positive changes in $w_-$ decrease $w$. We implement a small weight decay on the both synapses. Results from this circuit are shown in Figure 4(b).

### 3.3   Clustering Primitive

We tested a new clustering primitive that is based on the adaptive bump circuit introduced in [10]. The circuit performs two functions: 1) computes the similarity between an input and a stored value, and 2) adapts the stored value to decrease its distance to the input. This adaptive bump circuit exhibits improved adaptation over previous versions [10, 13] due to the inclusion of the autonulling differential pair[14], shown in Figure 5(a) (top). The autonulling differential pair ensures that the adaptation process increases the similarity between the stored mean and the input. The data in Figure 5(b) shows the clustering primitive adapting to an input that is initially distant from the stored value. The result of this adaptation is that over time, the circuit learns to produce a maximal output response at the present input.

This circuit was easily prototyped in the FPLA. Creation of a configuration file took less than one hour, experimental setup took another hour, and data was produced within two additional hours. Instead of waiting several months for chip fabrication, we were able to produce experimental results from a chip in under four hours. Also, the results are a more accurate model of actual circuit behavior than a SPICE simulation.

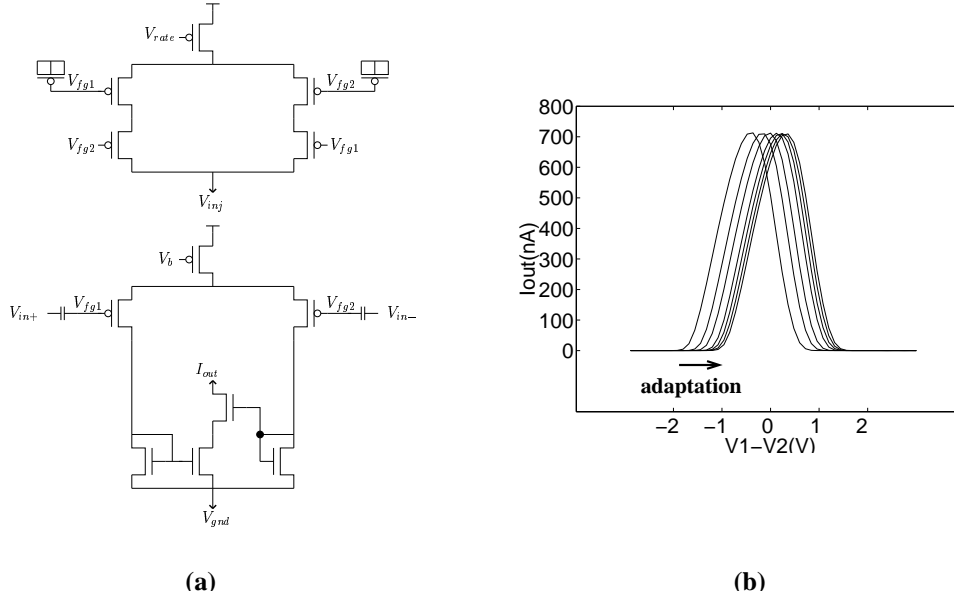

**(a)**                                                                 **(b)**

Figure 5: (a) Clustering Primitive. This circuit can: 1) compute the similarity between the stored value and the input, and 2) adapt the stored value to decrease its distance to the input. (b) Experimental Data. This plot shows that circuit adaptation moves the circuit's peak response toward the presented input. Adaptation strength decreases as the stored value approaches the input.

## 4  Future Work

The chip that we developed is effective for prototyping single learning primitives, but is too small for solving real machine-learning problems. An FPLA whose target is machine-learning algorithms requires PLBs that comprise higher-level functions, such as the primitives presented in the previous section.

To scale up our design for machine-learning applications, we will make the following improvements to our prototype. First, to reduce the size of the PLBs, we will increase the ratio of computational circuitry to switching circuitry by replacing the low-level functions such as current mirrors and synapse transistors with higher-level primitives such as those mentioned in the previous section. Second, we will increase the number of PLBs in the design, which will require an efficient and scalable global interconnect structure. We will base our revisions on commercial FPGA architectures and other well-known on-chip communication schemes. Third, we will improve the I/O structures to enable multichip systems. Finally, we have begun work on the design compiler, a software tool that maps machine-learning algorithms to an FPLA configuration.

## 5  Conclusions

Because of the match between the parallelism offered by hardware and the parallelism in machine-learning algorithms, mixed analog-digital VLSI is a promising substrate for machine-learning implementations. However, custom VLSI solutions are costly, inflexible, and difficult to design. To overcome these limitations, we have proposed Field-Programmable Learning Arrays, a viable reconfigurable architecture for prototyping machine-learning algorithms in hardware. FPLAs combine elements of FPGAs, analog VLSI, and on-chip learning to provide a scalable and cost-effective solution for learning

in silicon. Our results show that our prototype core and interconnect can effectively implement existing learning primitives and assist in the development of new circuits. An enhanced version of the FPLA, currently under development, will support complex learning algorithms.

**Acknowledgments**

This work was supported by ONR grant #N00014-01-1-0566 and an Intel Fellowship. Chips were fabricated by the MOSIS service.

# References

[1] C. Diorio, D. Hsu, and M. Figueroa, "Adaptive CMOS: From biological inspiration to systems-on-a-chip," *Proceedings of the IEEE*, vol. 90, no. 3, pp. 245–357, 2002.

[2] J. B. Burr, "Digital Neural Network Implementations," in *Neural Networks: Concepts, Applications, and Implementations, Volume 2* (P. Antognetti and V. Milutinovic, eds.), pp. 237–285, Prentice Hall, 1991.

[3] S. Satyanarayana, Y. Tsividis, and H. Graf, "A reconfigurable VLSI neural network," *IEEE Journal of Solid-State Circuits*, vol. 27, January 1992.

[4] R. Coggins, M. Jabri, B. Flower, and S. Pickard, "ICEG morphology classification using an analogue VLSI neural network," in *Advances in Neural Information Processing Systems 7*, pp. 731–738, MIT Press, 1995.

[5] M. Holler, S. Tam, H. Castro, and R. Benson, "An electrically trainable artificial neural network with 10240 'floating gate' synapses," in *Proceedings of the International Joint Conference on Neural Networks(IJCNN89)*, vol. 2, (Washington D.C), pp. 191–196, 1989.

[6] E. K. F. Lee and P. G. Gulak, "A CMOS field programmable analog array," *IEEE Journal of Solid-State Circuits*, vol. 26, December 1991.

[7] A. Montalvo, R. Gyurcsik, and J. Paulos, "An analog VLSI neural network with on-chip learning," *IEEE Journal of Solid-State Circuits*, vol. 32, no. 4, 1997.

[8] R. Genov and G. Cauwenberghs, "Stochastic mixed-signal VLSI architecture for high-dimensional kernel machines," in *Advances in Neural Information Processing Systems 14* (T. G. Dietterich, S. Becker, and Z. Ghahramani, eds.), (Cambridge, MA), MIT Press, 2002.

[9] J. Hyde, T. Humes, C. Diorio, M. Thomas, and M. Figueroa, "A floating-gate trimmed, 14-bit, 250 ms/s digital-to-analog converter in standard $0.25\mu$m CMOS," in *Symposium on VLSI Circuits Digest of Technical Papers*, pp. 328–331, 2002.

[10] D. Hsu, M. Figueroa, and C. Diorio, "A silicon primitive for competitive learning," in *Advances in Neural Information Processing Systems 13* (T. K. Leen, T. G. Dietterich, and V. Tresp, eds.), pp. 713–719, MIT Press, 2001.

[11] A. P. Shon, D. Hsu, and C. Diorio, "Learning spike-based correlations and conditional probabilities in silicon," in *Advances in Neural Information Processing Systems 14* (T. G. Dietterich, S. Becker, and Z. Ghahramani, eds.), (Cambridge, MA), MIT Press, 2002.

[12] C. Mead, *Analog VLSI and Neural Systems*. Reading, MA: Addison-Wesley, 1989.

[13] P. Hasler, "Continuous-time feedback in floating-gate MOS circuits," *IEEE Transactions on Circuits and Systems II*, vol. 48, pp. 56–64, January 2001.

[14] D. Hsu, S. Bridges, and C. Diorio, "Adaptive quantization and density estimation in silicon," 2002. In submission.
